# The Variational Ising Classifier (VIC) algorithm for coherently contaminated data

**Oliver Williams**
Dept. of Engineering
University of Cambridge
omcw2@cam.ac.uk

**Andrew Blake**
Microsoft Research Ltd.
Cambridge, UK

**Roberto Cipolla**
Dept. of Engineering
University of Cambridge

## Abstract

There has been substantial progress in the past decade in the development of object classifiers for images, for example of faces, humans and vehicles. Here we address the problem of contaminations (e.g. occlusion, shadows) in test images which have not explicitly been encountered in training data. The Variational Ising Classifier (VIC) algorithm models contamination as a mask (a field of binary variables) with a strong spatial coherence prior. Variational inference is used to marginalize over contamination and obtain robust classification. In this way the VIC approach can turn a kernel classifier for clean data into one that can tolerate contamination, without any specific training on contaminated positives.

## 1 Introduction

Recent progress in discriminative object detection, especially for faces, has yielded good performance and efficiency [1, 2, 3, 4]. Such systems are capable of classifying those positives that can be generalized from positive training data. This is restrictive in practice in that test data may contain distortions that take it outside the strict ambit of the training positives. One example would be lighting changes (to a face) but this can be addressed reasonably effectively by a normalizing transformation applied to training and test images; doing so is common practice in face classification. Other sorts of disruption are not so easily factored out. A prime example is partial occlusion.

The aim of this paper is to extend a classifier trained on clean positives to accept also partially occluded positives, without further training. The approach is to capture some of the regularity inherent in a typical pattern of contamination, namely its spatial coherence. This can be thought of as extending the generalizing capability of a classifier to tolerate the sorts of image distortion that occur as a result of contamination.

As done previously in one-dimension, for image contours [5], the Variational Ising Classifier (VIC) models contamination explicitly as switches with a strong coherence prior in the form of an Ising model, but here over the full two-dimensional image array. In addition, the Ising model is loaded with a bias towards non-contamination. The aim is to incorporate these hidden contamination variables into a kernel classifier such as [1, 3]. In fact the Relevance Vector Machine (RVM) is particularly suitable [6] as it is explicitly probabilistic, so that contamination variables can be incorporated as a hidden layer of random variables.

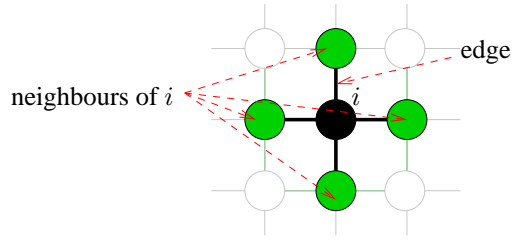

Figure 1: The 2D Ising model is applied over a graph with edges $e \in \Upsilon$ between neighbouring pixels (connected 4-wise).

Classification is done by marginalization over all possible configurations of the hidden variable array, and this is made tractable by variational (mean field) inference. The inference scheme makes use of "hallucination" to fill in parts of the object that are unobserved due to occlusion.

Results of VIC are given for face detection. First we show that the classifier performance is not significantly damaged by the inclusion of contamination variables. Then a contaminated test set is generated using real test images and computer generated contaminations. Over this test data the VIC algorithm does indeed perform significantly better than a conventional classifier (similar to [4]). The hidden variable layer is shown to operate effectively, successfully inferring areas of contamination. Finally, inference of contamination is shown working on real images with real contaminations.

## 2   Bayesian modelling of contamination

Classification requires $P(F|I)$, the posterior for the proposition $F$ that an object is present given the image data intensity array $I$. This can be computed in terms of likelihoods

$$P(F \mid I) = P(I \mid F)P(F)/\left(P(I \mid F)P(F) + P(I \mid \overline{F})P(\overline{F})\right) \qquad (1)$$

so then the test $P(F \mid I) > \frac{1}{2}$ becomes

$$\log P(I \mid F) - \log P(I \mid \overline{F}) > t \qquad (2)$$

where $t$ is a prior-dependent threshold that controls the tradeoff between positive and negative classification errors. Suppose we are given a likelihood $P(I|\theta, F)$ for the presence of a face given contamination $\theta$, an array of binary "observation" variables corresponding to each pixel $I_j$ of $I$, such that $\theta_j = 0$ indicates contamination at that pixel, whereas $\theta_j = 1$ indicates a successfully observed pixel. Then, in principle,

$$P(I|F) = \sum_{\theta} P(I|\theta, F)P(\theta), \qquad (3)$$

(making the reasonable assumption $P(\theta|F) = P(\theta)$, that the pattern of contamination is object independent) and similarly for $\log P(I \mid \overline{F})$. The marginalization itself is intractable, requiring a summation over all $2^N$ possible configurations of $\theta$, for images with $N$ pixels. Approximating that marginalization is dealt with in the next section. In the meantime, there are two other problems to deal with: specifying the prior $P(\theta)$; and specifying the likelihood under contamination $P(I|\theta, F)$ given only training data for the unoccluded object.

### 2.1   Prior over contaminations

The prior contains two terms: the first expresses the belief that contamination will occur in coherent regions of a subimage. This takes the form of an Ising model [7] with energy

$U_I(\theta)$ that penalizes adjacent pixels which differ in their labelling (see Figure 1); the second term $U_C$ biases generally against contamination *a priori* and its balance with the first term is mediated by the constant $\lambda$. The total prior energy is then

$$U(\theta) = U_I(\theta) + \lambda U_C(\theta) = \sum_{e \in \Upsilon}[1 - \delta(\theta_{e_1} - \theta_{e_2})] + \lambda \sum_j \delta(\theta_j), \tag{4}$$

where $\delta(x) = 1$ if $x = 0$ and 0 otherwise, and $e_1$, $e_2$ are the indices of the pixels at either end of edge $e \in \Upsilon$ (figure 1). The prior energy determines a probability via a *temperature* constant $1/T_0$ [7]:

$$P(\theta) \propto e^{-U(\theta)/T_0} = e^{-U_I(\theta)/T_0} e^{-\lambda U_C(\theta)/T_0} \tag{5}$$

## 2.2 Relevance vector machine

An unoccluded classifier $P(F|I, \theta = \mathbf{0})$ can be learned from training data using a Relevance Vector Machine (RVM) [6], trained on a database of *frontal* face and non-face images [8] (see Section 4 for details). The probabilistic properties of the RVM make it a good choice when (later) it comes to marginalising over $\theta$. For now we consider how to construct the likelihood itself. First the conventional, unoccluded case is considered for which the posterior $P(F|I)$ is learned from positive and negative examples. Kernel functions [9] are computed between a candidate image $I$ and a subset of *relevance vectors* $\{x_k\}$, retained from the training set. Gaussian kernels are used here to compute

$$y(I) = \sum_k w_k \exp\left(-\alpha \sum_j (I_j - x_{kj})^2\right). \tag{6}$$

where $w_k$ are learned weights, and $x_{kj}$ is the $j^{\text{th}}$ pixel of the $k^{\text{th}}$ relevance vector. Then the posterior is computed via the logistic sigmoid function as

$$P(F|I, \theta = \mathbf{1}) = \sigma(y(I)) = \frac{1}{1 + e^{-y(I)}}. \tag{7}$$

and finally the unoccluded data-likelihood would be

$$P(I|F, \theta = \mathbf{1}) \propto \sigma(y(I))/P(F). \tag{8}$$

## 2.3 Hallucinating appearance

The aim now is to derive the occluded likelihood from the unoccluded case, where the contamination mask is known, without any further training. To do this, (8) must be extended to give $P(I|F, \theta)$ for arbitrary masks $\theta$, despite the fact the pixels $I_j$ from the object are not observed wherever $\theta_j = 0$. In principle one should take into account all possible (or at least probable) values for the occluded pixels. Here, for simplicity, a single fixed hallucination is substituted for occluded pixels, then we proceed as if those values had actually been observed. This gives

$$P(I|F, \theta) \propto \sigma(\tilde{y}(I, \theta))/P(F) \tag{9}$$

where

$$\tilde{y}(\theta, I) = y(\tilde{I}(I, \theta, F)) \text{ and } \left(\tilde{I}(I, \theta, F)\right)_j = \begin{cases} I_j & \text{if } \theta_j = 1 \\ (\mathcal{E}[I|F])_j & \text{otherwise} \end{cases} \tag{10}$$

in which $\mathcal{E}[I|F]$ is a fixed hallucination, conditioned on the model $F$, and computed as a sample mean over training instances.

# 3 Approximate marginalization of $\theta$ by mean field

At this point we return to the task of marginalising over $\theta$ (3) to obtain $P(I|F)$ and $P(I|\overline{F})$ for use in classification (2). Due to the connectedness of neighbouring pixels in the Ising prior (figure 1), $P(I, \theta|F)$ is a Markov Random Field (MRF) [7]. The marginalized likelihood $P(I|F)$ could be estimated by Gibbs sampling [10] but that takes tens of minutes to converge in our experiments. The following section describes a mean field approximation which converges in a few seconds. The mean field algorithm is given here for $P(I|F)$ but must be repeated also for $P(I|\overline{F})$, simply substituting $\overline{F}$ for $F$ throughout.

## 3.1 Variational approximation

Mean field approximation is a form of variational approximation [11] and transforms an inference problem into the optimization of a functional $J$:

$$J(Q) = \log P(I|F) - \text{KL}\left[Q(\theta)\|P(\theta|F, I)\right], \tag{11}$$

where KL is the Kullback-Liebler divergence

$$\text{KL}\left[Q(\theta)\|P(\theta|F, I)\right] = \sum_\theta Q(\theta) \log \frac{Q(\theta)}{P(\theta|F, I)}.$$

The objective functional $J(Q)$ is a lower bound on the log-marginal probability $\log P(I|F)$ [11]; when it is maximized at $Q^*$, it gives both the marginal likelihood $J(Q^*) = \log P(I|F)$, and the posterior distribution $Q^*(\theta) = P(\theta|F, I)$ over hidden variables. Following [11], $J(Q)$ is simplified using Bayes' rule:

$$J(Q) = H(Q) + \mathcal{E}_Q\left[\log P(I, \theta|F)\right]$$

where $H(\cdot)$ is the entropy of a distribution [12] and $\mathcal{E}_Q[g(\theta)] = \sum_\theta Q(\theta)g(\theta)$ denotes the expectation of a function $g$ with respect to $Q(\theta)$. A form of $Q(\theta)$ must be chosen that makes the maximization of $J(Q)$ tractable. For mean-field approximation, $Q(\theta)$ is modelled as a pixel-wise product of factors: $Q(\theta) = \prod_i Q_i(\theta_i)$. It is now possible to maximize $J$ iteratively with respect to each marginal $Q_i(\theta_i)$ in turn, giving the *mean field update* [11]:

$$Q_i \leftarrow \frac{1}{Z_i} \exp\left\{\mathcal{E}_{Q|\theta_i}\left[\log P(I, \theta|F)\right]\right\}, \tag{12}$$

where

$$Z_i = \sum_{\theta_i} \exp\left\{\mathcal{E}_{Q|\theta_i}\left[\log P(I, \theta|F)\right]\right\}$$

is the partition function and $\mathcal{E}_{Q|\theta_i}[\cdot]$ is the expectation with respect to $Q$ given $\theta_i$:

$$\mathcal{E}_{Q|\theta_i}[g(\theta)] = \sum_{\{\theta\}_{j\backslash i}} \left[\prod_{j\backslash i} Q_j(\theta_j)\right] g(\theta).$$

## 3.2 Taking expectations over $P(I, \theta|F)$

To perform the expectation required in (12), the log-joint distribution is written as:

$$\log\left\{P(I, \theta|F)\right\} = -\log\left(1 + e^{-\tilde{y}(\theta, I)}\right) - \tfrac{1}{T_0}U_I(\theta) - \tfrac{\lambda}{T_0}U_C(\theta) + \text{const.}$$

The conditional expectation $\mathcal{E}_{Q|\theta_i}$ in (12) is found efficiently from the complete expectations by replacing only terms in $\theta_i$. Likewise, when one factor of $Q$ changes (12), the

complete expectations may be updated without recomputing them *ab initio*. For brevity, we give the expressions for the complete expectations only. For the prior this is simply:

$$\mathcal{E}_Q[U(\theta)] = \sum_{e \in \Upsilon} \sum_{\theta_e} Q_e(\theta_e) \left[1 - \delta(\theta_{e_1} - \theta_{e_2})\right] + \lambda \sum_j Q_j(\theta_j = 0). \tag{13}$$

For the likelihood it is more difficult. Saul et al. [13] show how to approximate the expectation over the sigmoid function by introducing a dummy variable $\xi$:

$$\mathcal{E}_Q \left[\log(1 + e^{-\tilde{y}(\theta,I)})\right] \leq -\xi \mathcal{E}_Q[\tilde{y}(\theta,I)] + \log \left\{ \mathcal{E}_Q \left[ e^{\xi \tilde{y}(\theta,I)} \right] + \mathcal{E}_Q \left[ e^{(\xi-1)\tilde{y}(\theta,I)} \right] \right\}.$$

The Gaussian RBF in (6) means that it is not feasible to compute the expectation[1] $\mathcal{E}_Q \left[ e^{\xi \tilde{y}(\theta,I)} \right]$, so a simpler approximation is used:

$$\mathcal{E}_Q[\log \sigma(\tilde{y}(\theta,I)] \approx \log \sigma \left( \mathcal{E}_Q[\tilde{y}(\theta,I)] \right),$$

where

$$\mathcal{E}_Q[\tilde{y}(\theta,I)] = \sum_k w_k \prod_j \sum_{\theta_j} Q_j(\theta_j) \exp\left(-\alpha \big(\tilde{I}(I,\theta,F)_j - x_{kj}\big)^2\right). \tag{14}$$

## 4  Results and discussion

The mean field algorithm described above is capable only of local optimization of $J(Q)$. A symptom of this is that it exhibits *spontaneous symmetry breaking* [11], setting the contamination field to either all contaminated or all uncontaminated. This is alleviated through careful initialization. By performing iterations initially at a high temperature, $T_h$, the prior is weakened. The temperature is then progressively decreased, on a linear annealing schedule [10], until the modelled prior temperature $T_0$ is reached. Figure 2 shows pseudo-code for the VIC algorithm. Note also that an advantage of hallucinating appearance from the mean face is that the hallucination process requires no computation within the optimization loop. For $19 \times 19$ subimages, the average time taken for the VIC algorithm to converge is 4 seconds. However this is an unoptimized Matlab implementation; and in C++ it is anticipated to be at least 10 times faster.

The training set used for the RVM [8] contains subimages of registered faces and non-faces which were histogram equalized [14] to reduce the effect of different lighting with their pixel values scaled to the range $[0, 1]$. The same is done to each test subimage $I$. The RVM was trained using 1500 face examples and 1500 non-face examples [2]. Parameters were set as follows: the RBF width parameter in (6) is $\alpha = 0.05$; the contamination cost $\lambda = 0.2$ and the temperature constants are $T_h = 2.5$, $T_0 = 1.5$ and $\Delta T = 0.2$.

As a by-product of the VIC algorithm, the posterior pattern $P(\theta|F, I)$ of contamination is approximately inferred as the value of $Q$ which maximizes $J$. Figure 3 shows some results of this. As might be expected, for a non-face, the algorithm hallucinates an intact face with total contamination (For example, row 4 of the figure); but of course the marginalized posterior probability $P(F|I)$ is very small in such a case.

### 4.1  Classifier

To assess the classification performance of the VIC, contaminated positives were automatically generated (figure 4). These were combined with pure faces and pure non-faces (none of which were used in the training set) and tested to produce the Receiver Operating Characteristic (ROC) curves are given in Figure 4 for the unaltered RVM acting on the

**Require:** Candidate image region $I$
**Require:** Parameters $T_h$, $T_0$, $\Delta T$, $\lambda$
**Require:** RVM weights and examples $w_k, x_k$
**Require:** Mean face appearance $\bar{I}$

Initialize $Q_i(\theta_i = 1) \leftarrow 0.5 \quad \forall i$
Compute $\mathcal{E}_Q[U(\theta)]$ (13)
Compute $\mathcal{E}_Q[\tilde{y}(\theta, I)]$ (14)

$T \leftarrow T_h$
**while** $T > T_0$ **do**
  **while** $Q$ not converged **do**
    **for** All image locations $i$ **do**
      Compute conditional expectations $\mathcal{E}_{Q|\theta_i}[U(\theta)]$ and $\mathcal{E}_{Q|\theta_i}[\tilde{y}(\theta, I)]$
      Compute $\mathcal{E}_{Q|\theta_i}[\log P(I, \theta|F)] = \log \sigma\big(\mathcal{E}_{Q|\theta_i}[\tilde{y}(\theta, I)]\big) - \mathcal{E}_{Q|\theta_i}[U(\theta)]$
      Compute partition $Z_i = \sum_{\theta_i} \exp\big\{\mathcal{E}_{Q|\theta_i}[\log P(I, \theta|F)]\big\}$
      Update $Q_i(\theta_i) \leftarrow \frac{1}{Z_i} \exp\big\{\mathcal{E}_{Q|\theta_i}[\log P(I, \theta|F)]\big\}$
      Update complete expectations $\mathcal{E}_Q[U(\theta)]$ and $\mathcal{E}_Q[\tilde{y}(\theta, I)]$
    **end for**
    $T \leftarrow T - \Delta T$
  **end while**
**end while**

Figure 2: Pseudo-code for the VIC algorithm

Input $I$     Hallucinated image     Contamination field $Q(\theta = 1)$

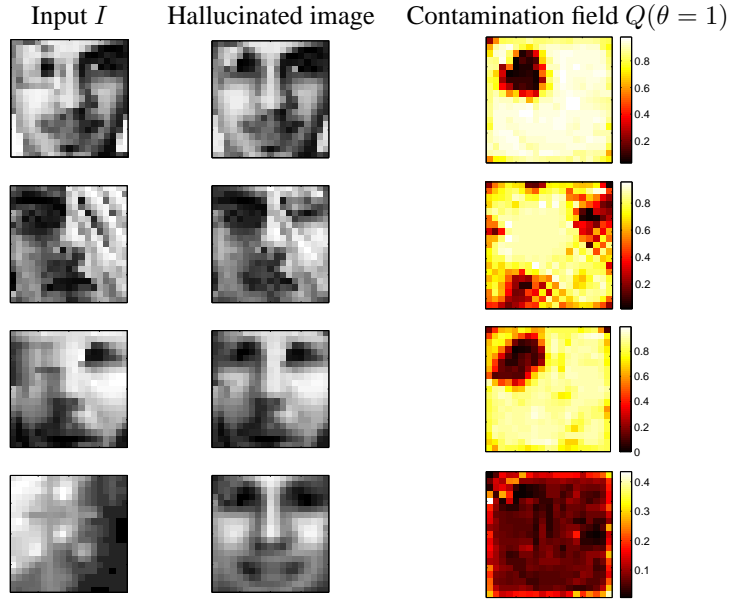

Figure 3: Partially occluded mages with inferred areas of probable contamination (dark).

contaminated set and for the new contamination-tolerant VIC outlined in this paper. For comparison, points are shown for a *boosted cascade of classifiers* [15] which is a publicly available detector based on the system of Viola and Jones [4]. The curve shown for the RVM against an *uncontaminated* test set confirms that contamination does make the classification task considerably harder. Figure 5 shows some natural face images that the boosted cascade [15] fails to detect, either because of occlusion or due to a degree of deviation from

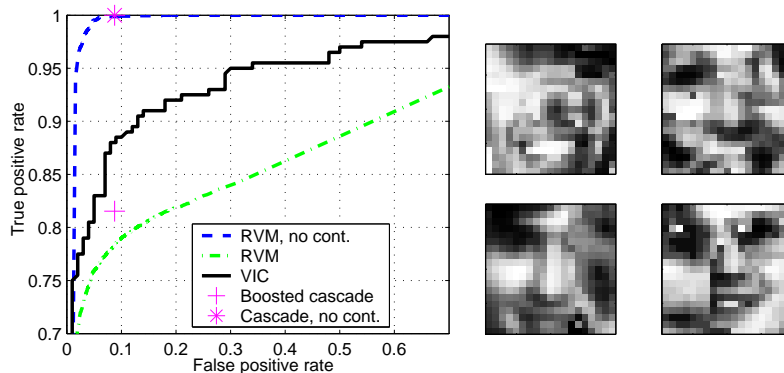

Figure 4: ROC curves. Also shown are some of the contaminated positives used to generate the curves. These were made by sampling contamination patterns from the prior and using them to mix a face and a non-face artificially.

| Input $I$ | Hallucinated image | Contamination field $Q(\theta = 1)$ |

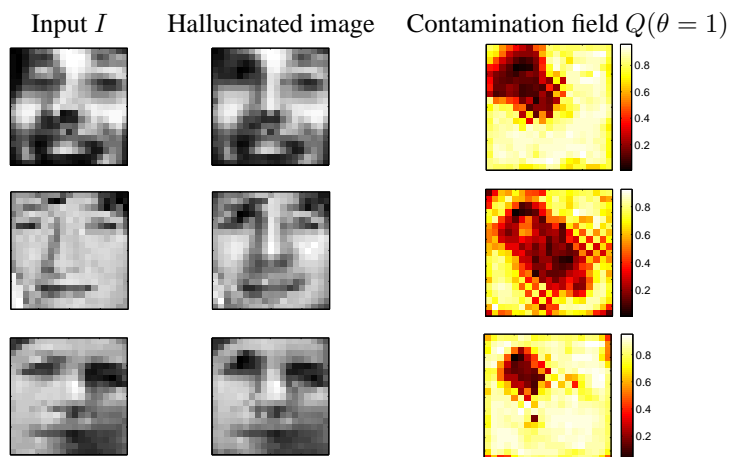

Figure 5: Images that the boosted cascade [15] failed to detect as faces: the VIC algorithm produces higher posterior face probability by labelling certain regions with unusual appearance (eg due to 3D rotation) as contaminated.

the frontal pose. The VIC algorithm detects them successfully however.

## 4.2 Discussion

Figure 4 shows that by modelling the contamination field explicitly, the VIC detector improves on the performance, over a contaminated test set, both of a plain RVM and of a boosted cascade detector. The algorithm is relatively expensive to execute compared, say, with the contamination-free RVM. However, this could be mitigated by cascading [4], in which a simple and efficient classifier, tuned to return a high rate of false positives for all objects, contaminated and non-contaminated, would make a preliminary sweep of a test image. The contamination-tolerant VIC algorithm would then be applied to the candidate subimages that remain, thereby concentrating computational power on just a few locations.

Figure 5 illustrates the operation of the contamination mechanism on real images, all of

which are detected as faces by the VIC algorithm but missed by the boosted cascade. There is no occlusion in these examples but rotations have distorted the appearance of certain features. The VIC algorithm has deals with this by labelling the distortions as contaminated areas, and hallucinating face-like texture in their place.

In conclusion, we have developed the VNC algorithm for object detection in the presence of coherently contaminated data. Contamination is modelled as coherent via an Ising prior, and is marginalized out by variational inference. Experiments show that VIC classifies contaminated images more robustly than classifiers designed for clean data. It is worth pointing out that the approach of the VIC algorithm is not limited to RVMs. Any probabilistic detector for which it is possible to estimate the expectation (14) could be modified in a similar way to deal with spatially coherent contamination. Future work will address: improved efficiency by incorporating the VIC into a cascade of simple classifiers; alternatives to data hallucination using marginalization over missing data, if a tractable means of doing this can be found.

## Footnotes

[1]The term $\exp[\xi \tilde{y}(\theta,I)] = \exp[\xi \sum_k w_k \prod_j e^{-\alpha d_j(I, x_k|\theta_j)}]$ does not factorize across pixels

[2]These sizes are limited in practice by the complexity of the training algorithm [6]

# References

[1] E. Osuna, R. Freund, and F. Girosi. Training support vector machines: An application to face detection. *Proc. Conf. Computer Vision and Pattern Recognition*, pages 130–136, 1997.

[2] H.A. Rowley, S. Baluja, and T. Kanade. Neural network-based face detection. *IEEE Transactions on Pattern Alaysis and Machine Intelligence*, 20(1):23–38, 1998.

[3] S. Romdhani, P. Torr, B. Schölkopf, and A. Blake. Computationally efficient face detection. In *Proc. Int. Conf. on Computer Vision*, volume 2, pages 524–531, 2001.

[4] P. Viola and M. Jones. Rapid object detection using a boosted cascade of simple features. In *Proc. Conf. Computer Vision and Pattern Recognition*, 2001.

[5] J. MacCormick and A. Blake. Spatial dependence in the observation of visual contours. In *Proc. European Conf. on Computer Vision*, pages 765–781, 1998.

[6] M.E. Tipping. Sparse Bayesian learning and the relevance vector machine. *Journal of Machine Learning Research*, 1:211–244, 2001.

[7] R. Kindermann and J.L. Snell. *Markov Random Fields and Their Applications*. American Mathematical Society, 1980.

[8] CBCL face database #1. MIT Center For Biological and Computation Learning: http://www.ai.mit.edu/projects/cbcl.

[9] B. Schölkopf and A. Smola. *Learning with Kernels: Support Vector Machines, Regularization, Optimization, and Beyond (Adaptive Computation and Machine Learning)*. MIT Press, 2001.

[10] S. Geman and D. Geman. Stochastic relaxation, Gibbs distributions, and the Bayesian restoration of images. *IEEE Trans. on Pattern Analysis and Machine Intelligence*, 6(6):721–741, 1984.

[11] T. Jaakkola. Tutorial on variational approximation methods. In *Advanced Mean Field Methods: Theory and Practice*. MIT Press, 2000.

[12] T. Cover and J. Thomas. *Elements of Information Theory*. John Wiley & Sons, 1991.

[13] L. Saul, T. Jaakkola, and M. Jordan. Mean field theory for sigmoid belief networks. *Journal of Artificial Intelligence Research*, 4:61–76, 1996.

[14] A.K. Jain. *Fundamentals of Digital Image Processing*. System Sciences. Prentice-Hall, New Jersey, 1989.

[15] R. Lienhart and J. Maydt. An extended set of Haar-like features for rapid object detection. In *Proc. IEEE ICIP*, volume 1, pages 900–903, 2002.
